# Comparing Bayesian models for multisensory cue combination without mandatory integration

**Ulrik R. Beierholm**
Computation and Neural Systems
California Institute of Technology
Pasadena, CA 91025
beierh@caltech.edu

**Konrad P. Körding**
Rehabilitation Institute of Chicago
Northwestern University, Dept. PM&R
Chicago, IL 60611
konrad@koerding.com

**Ladan Shams**
Department of Psychology
University of California, Los Angeles
Los Angeles, CA 90095
ladan@psych.ucla.edu

**Wei Ji Ma**
Department of Brain and Cognitive Sciences
University of Rochester
Rochester, NY 14620
weijima@gmail.com

## Abstract

Bayesian models of multisensory perception traditionally address the problem of estimating an underlying variable that is assumed to be the cause of the two sensory signals. The brain, however, has to solve a more general problem: it also has to establish which signals come from the same source and should be integrated, and which ones do not and should be segregated. In the last couple of years, a few models have been proposed to solve this problem in a Bayesian fashion. One of these has the strength that it formalizes the causal structure of sensory signals. We first compare these models on a formal level. Furthermore, we conduct a psychophysics experiment to test human performance in an auditory-visual spatial localization task in which integration is not mandatory. We find that the causal Bayesian inference model accounts for the data better than other models.

Keywords: causal inference, Bayesian methods, visual perception.

## 1   Multisensory perception

In the ventriloquist illusion, a performer speaks without moving his/her mouth while moving a puppet's mouth in synchrony with his/her speech. This makes the puppet appear to be speaking. This illusion was first conceptualized as "visual capture", occurring when visual and auditory stimuli exhibit a small conflict ([1, 2]). Only recently has it been demonstrated that the phenomenon may be seen as a byproduct of a much more flexible and nearly Bayes-optimal strategy ([3]), and therefore is part of a large collection of cue combination experiments showing such statistical near-optimality [4, 5]. In fact, cue combination has become the poster child for Bayesian inference in the nervous system.

In previous studies of multisensory integration, two sensory stimuli are presented which act as cues about a single underlying source. For instance, in the auditory-visual localization experiment by Alais and Burr [3], observers were asked to envisage each presentation of a light blob and a sound click as a single event, like a ball hitting the screen. In many cases, however, the brain is not only posed with the problem of identifying the position of a common source, but also of determining whether there was a common source at all. In the on-stage ventriloquist illusion, it is indeed primarily the causal inference process that is being fooled, because veridical perception would attribute independent causes to the auditory and the visual stimulus.

To extend our understanding of multisensory perception to this more general problem, it is necessary to manipulate the degree of belief assigned to there being a common cause within a multisensory task. Intuitively, we expect that when two signals are very different, they are less likely to be perceived as having a common source. It is well-known that increasing the discrepancy or inconsistency between stimuli reduces the influence that they have on each other [6, 7, 8, 9, 10, 11]. In auditory-visual spatial localization, one variable that controls stimulus similarity is spatial disparity (another would be temporal disparity). Indeed, it has been reported that increasing spatial disparity leads to a decrease in auditory localization bias [1, 12, 13, 14, 15, 16, 17, 2, 18, 19, 20, 21]. This decrease also correlates with a decrease in the reports of unity [19, 21]. Despite the abundance of experimental data on this issue, no general theory exists that can explain multisensory perception across a wide range of cue conflicts.

## 2 Models

The success of Bayesian models for cue integration has motivated attempts to extend them to situations of large sensory conflict and a consequent low degree of integration. In one of recent studies taking this approach, subjects were presented with concurrent visual flashes and auditory beeps and asked to count both the number of flashes and the number of beeps [11]. The advantage of the experimental paradigm adopted here was that it probed the joint response distribution by requiring a dual report. Human data were accounted for well by a Bayesian model in which the joint prior distribution over visual and auditory number was approximated from the data. In a similar study, subjects were presented with concurrent flashes and taps and asked to count either the flashes or the taps [9, 22]. The Bayesian model proposed by these authors assumed a joint prior distribution with a near-diagonal form. The corresponding generative model assumes that the sensory sources somehow interact with one another. A third experiment modulated the rates of flashes and beeps. The task was to judge either the visual or the auditory modulation rate relative to a standard [23]. The data from this experiment were modeled using a joint prior distribution which is the sum of a near-diagonal prior and a flat background.

While all these models are Bayesian in a formal sense, their underlying generative model does not formalize the model selection process that underlies the combination of cues. This makes it necessary to either estimate an empirical prior [11] by fitting it to human behavior or to assume an ad hoc form [22, 23]. However, we believe that such assumptions are not needed. It was shown recently that human judgments of spatial unity in an auditory-visual spatial localization task can be described using a Bayesian inference model that infers causal structure [24, 25]. In this model, the brain does not only estimate a stimulus variable, but also infers the probability that the two stimuli have a common cause. In this paper we compare these different models on a large data set of human position estimates in an auditory-visual task.

In this section we first describe the traditional cue integration model, then the recent models based on joint stimulus priors, and finally the causal inference model. To relate to the experiment in the next section, we will use the terminology of auditory-visual spatial localization, but the formalism is very general.

### 2.1 Traditional cue integration

The traditional generative model of cue integration [26] has a single source location $s$ which produces on each trial an internal representation (cue) of visual location, $x_V$ and one of auditory location, $x_A$. We assume that the noise processes by which these internal representations are generated are conditionally independent from each other and follow Gaussian distributions. That is, $p(x_V|s) \sim N(x_V; s, \sigma_V)$ and $p(x_A|s) \sim N(x_A; s, \sigma_A)$, where $N(x; \mu, \sigma)$ stands for the normal distribution over $x$ with mean $\mu$ and standard deviation $\sigma$. If on a given trial the internal representations are $x_V$ and $x_A$, the probability that their source was $s$ is given by Bayes' rule,

$$p(s|x_V, x_A) \propto p(x_V|s) p(x_A|s).$$

If a subject performs maximum-likelihood estimation, then the estimate will be

$\hat{s} = \frac{w_V x_V + w_A x_A}{w_V + w_A}$, where $w_V = \frac{1}{\sigma_V^2}$ and $w_A = \frac{1}{\sigma_A^2}$. It is important to keep in mind that this is the estimate on a single trial. A psychophysical experimenter can never have access to $x_V$ and $x_A$, which

are the noisy internal representations. Instead, an experimenter will want to collect estimates over many trials and is interested in the distribution of $\hat{s}$ given $s_V$ and $s_A$, which are the sources generated by the experimenter. In a typical cue combination experiment, $x_V$ and $x_A$ are not actually generated by the same source, but by different sources, a visual one $s_V$ and an auditory one $s_A$. These sources are chosen close to each other so that the subject can imagine that the resulting cues originate from a single source and thus implicitly have a common cause. The experimentally observed distribution is then

$$p\left(\hat{s}|s_V, s_A\right) = \int \int p\left(\hat{s}|x_V, x_A\right) p\left(x_V|s_V\right) p\left(x_A|s_A\right) dx_V dx_A$$

Given that $\hat{s}$ is a linear combination of two normally distributed variables, it will itself follow a normal distribution, with mean $\langle \hat{s} \rangle = \frac{w_V s_V + w_A s_A}{w_V + w_A}$ and variance $\sigma_{\hat{s}}^2 = \frac{1}{w_V + w_A}$. The reason that we emphasize this point is because many authors identify the estimate distribution $p\left(\hat{s}|s_V, s_A\right)$ with the posterior distribution $p\left(s|x_V, x_A\right)$. This is justified in this case because all distributions are Gaussian and the estimate is a linear combination of cues. However, in the case of causal inference, these conditions are violated and the estimate distribution will in general not be the same as the posterior distribution.

## 2.2 Models with bisensory stimulus priors

Models with bisensory stimulus priors propose the posterior over source positions to be proportional to the product of unimodal likelihoods and a two-dimensional prior:

$$p\left(s_V, s_A|x_V, x_A\right) = p\left(s_V, s_A\right) p\left(x_V|s_V\right) p\left(x_A|s_A\right)$$

The traditional cue combination model has $p\left(s_V, s_A\right) = p\left(s_V\right) \delta\left(s_V - s_A\right)$, usually (as above) even with $p\left(s_V\right)$ uniform. The question arises what bisensory stimulus prior is appropriate. In [11], the prior is estimated from data, has a large number of parameters, and is therefore limited in its predictive power. In [23], it has the form

$$p\left(s_V, s_A\right) \propto \omega + e^{-\frac{(s_V - s_A)^2}{2\sigma_{\text{coupling}}^2}}$$

while in [22] the additional assumption $\omega = 0$ is made[1].

In all three models, the response distribution $p\left(\hat{s}_V, \hat{s}_A|s_V, s_A\right)$ is obtained by identifying it with the posterior distribution $p\left(s_V, s_A|x_V, x_A\right)$. This procedure thus implicitly assumes that marginalizing over the latent variables $x_V$ and $x_A$ is not necessary, which leads to a significant error for non-Gaussian priors. In this paper we correctly deal with these issues and in all cases marginalize over the latent variables. The parametric models used for the coupling between the cues lead to an elegant low-dimensional model of cue integration that allows for estimates of single cues that differ from one another.

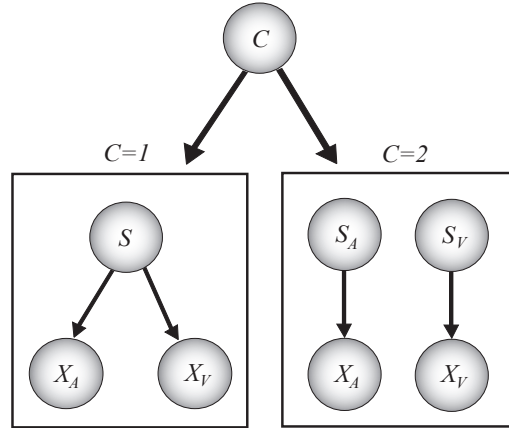

Figure 1: Generative model of causal inference.

## 2.3 Causal inference model

In the causal inference model [24, 25], we start from the traditional cue integration model but remove the assumption that two signals are caused by the same source. Instead, the number of sources can be one or two and is itself a variable that needs to be inferred from the cues.

If there are two sources, they are assumed to be independent. Thus, we use the graphical model depicted in Fig. 1. We denote the number of sources by $C$. The probability distribution over $C$ given internal representations $x_V$ and $x_A$ is given by Bayes' rule:

$$p\left(C|x_V, x_A\right) \propto p\left(x_V, x_A|C\right) p\left(C\right).$$

In this equation, $p\left(C\right)$ is the a priori probability of $C$. We will denote the probability of a common cause by $p_{\text{common}}$, so that $p\left(C = 1\right) = p_{\text{common}}$ and $p\left(C = 2\right) = 1 - p_{\text{common}}$. The probability of generating $x_V$ and $x_A$ given $C$ is obtained by inserting a summation over the sources:

$$p\left(x_V, x_A|C = 1\right) = \int p\left(x_V, x_A|s\right)p\left(s\right) ds = \int p\left(x_V|s\right) p\left(x_A|s\right)p\left(s\right) ds$$

Here $p\left(s\right)$ is a prior for spatial location, which we assume to be distributed as $N\left(s; 0, \sigma_P\right)$. Then all three factors in this integral are Gaussians, allowing for an analytic solution: $p\left(x_V, x_A|C = 1\right) = \frac{1}{2\pi\sqrt{\sigma_V^2\sigma_A^2 + \sigma_V^2\sigma_P^2 + \sigma_A^2\sigma_P^2}} \exp\left[-\frac{1}{2}\frac{(x_V - x_A)^2\sigma_P^2 + x_V^2\sigma_A^2 + x_A^2\sigma_V^2}{\sigma_V^2\sigma_A^2 + \sigma_V^2\sigma_P^2 + \sigma_A^2\sigma_P^2}\right]$.

For $p\left(x_V, x_A|C = 2\right)$ we realize that $x_V$ and $x_A$ are independent of each other and thus obtain

$$p\left(x_V, x_A|C = 2\right) = \left(\int p\left(x_V|s_V\right)p\left(s_V\right) ds_V\right)\left(\int p\left(x_A|s_A\right)p\left(s_A\right) ds_A\right)$$

Again, as all these distributions are assumed to be Gaussian, we obtain an analytic solution, $p\left(x_V, x_A|C = 2\right) = \frac{1}{2\pi\sqrt{\left(\sigma_V^2 + \sigma_P^2\right)\left(\sigma_A^2 + \sigma_P^2\right)}} \exp\left[-\frac{1}{2}\left(\frac{x_V^2}{\sigma_V^2 + \sigma_P^2} + \frac{x_A^2}{\sigma_A^2 + \sigma_P^2}\right)\right]$. Now that we have computed $p\left(C|x_V, x_A\right)$, the posterior distribution over sources is given by

$$p\left(s_i|x_V, x_A\right) = \sum_{C=1,2} p\left(s_i|x_V, x_A, C\right) p\left(C|x_V, x_A\right)$$

where $i$ can be V or A and the posteriors conditioned on $C$ are well-known:

$$p\left(s_i|x_A, x_V, C = 1\right) = \frac{p\left(x_A|s_i\right) p\left(x_V|s_i\right) p\left(s_i\right)}{\int p\left(x_A|s\right) p\left(x_V|s\right) p\left(s\right) ds}, \quad p\left(s_i|x_A, x_V, C = 2\right) = \frac{p\left(x_i|s_i\right) p\left(s_i\right)}{\int p\left(x_i|s_i\right) p\left(s_i\right) ds_i}$$

The former is the same as in the case of mandatory integration with a prior, the latter is simply the unimodal posterior in the presence of a prior. Based on the posterior distribution on a given trial, $p\left(s_i|x_V, x_A\right)$, an estimate has to be created. For this, we use a sum-squared-error cost function, $\text{Cost} = \left\langle p\left(C = 1|x_V, x_A\right)\left(\hat{s} - s\right)^2\right\rangle + \left\langle p\left(C = 2|x_V, x_A\right)\left(\hat{s} - s_{V \text{ or } A}\right)^2\right\rangle$. Then the best estimate is the mean of the posterior distribution, for instance for the visual estimation:

$$\hat{s}_V = p\left(C = 1|x_A, x_V\right) \hat{s}_{V,C=1} + p\left(C = 2|x_A, x_V\right) \hat{s}_{V,C=2}$$

where $\hat{s}_{V,C=1} = \frac{x_V\sigma_V^{-2} + x_A\sigma_A^{-2} + x_P\sigma_P^{-2}}{\sigma_V^{-2} + \sigma_A^{-2} + \sigma_P^{-2}}$ and $\hat{s}_{V,C=2} = \frac{x_V\sigma_V^{-2} + x_P\sigma_P^{-2}}{\sigma_V^{-2} + \sigma_P^{-2}}$. If $p_{\text{common}}$ equals 0 or 1, this estimate reduces to one of the conditioned estimates and is linear in $x_V$ and $x_A$. If $0 < p_{\text{common}} < 1$, the estimate is a *nonlinear* combination of $x_V$ and $x_A$, because of the functional form of $p\left(C|x_V, x_A\right)$. The response distributions, that is the distributions of $\hat{s}_V$ and $\hat{s}_A$ given $s_V$ and $s_A$ over many trials, now cannot be identified with the posterior distribution on a single trial and cannot be computed analytically either. The correct way to obtain the response distribution is to simulate an experiment numerically.

Note that the causal inference model above can also be cast in the form of a bisensory stimulus prior by integrating out the latent variable C, with:

$$p\left(s_A, s_V\right) = p\left(C = 1\right) \delta\left(s_A - s_V\right) p\left(s_A\right) + p\left(s_A\right) p\left(s_V\right) p\left(C = 2\right)$$

However, in addition to justifying the form of the interaction between the cues, the causal inference model has the advantage of being based on a generative model that well formalizes salient properties of the world, and it thereby also allows to predict judgments of unity.

# 3  Model performance and comparison

To examine the performance of the causal inference model and to compare it to previous models, we performed a human psychophysics experiment in which we adopted the same dual-report paradigm as was used in [11]. Observers were simultaneously presented with a brief visual and also an auditory stimulus, each of which could originate from one of five locations on an imaginary horizontal line (-10°, -5°, 0°, 5°, or 10° with respect to the fixation point). Auditory stimuli were 32 ms of white noise filtered through an individually calibrated head related transfer function (HRTF) and presented through a pair of headphones, whereas the visual stimuli were high contrast Gabors on a noisy background presented on a 21-inch CRT monitor. Observers had to report by means of a key press (1-5) the perceived positions of both the visual and the auditory stimulus. Each combination of locations was presented with the same frequency over the course of the experiment. In this way, for each condition, visual and auditory response histograms were obtained.

We obtained response distributions for each the three models described above by numeral simulation. On each trial, estimation is followed by a step in which, the key is selected which corresponds to the position closed to the best estimate. The simulated histograms obtained in this way were compared to the measured response frequencies of all subjects by computing the $R^2$ statistic.

The parameters in the causal inference model were optimized using fminsearch in MATLAB to maximize $R^2$. The best combination of parameters yielded an $R^2$ of 0.97. The response frequencies are depicted in Fig. 2. The bisensory prior models also explain most of the variance, with $R^2 = 0.96$ for the Roach model and $R^2 = 0.91$ for the Bresciani model. This shows that it is possible to model cue combination for large disparities well using such models.

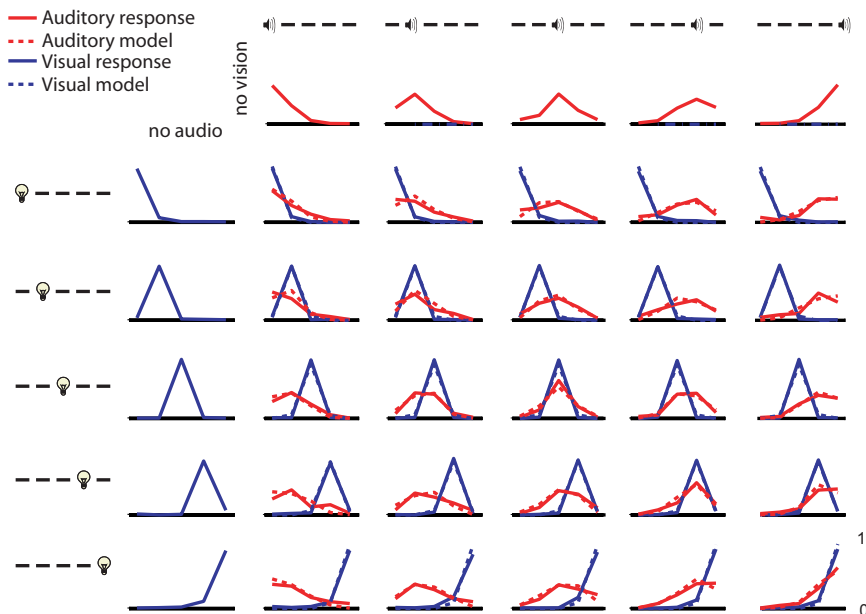

Figure 2: A comparison between subjects' performance and the causal inference model. The blue line indicates the frequency of subjects responses to visual stimuli, red line is the responses to auditory stimuli. Each set of lines is one set of audio-visual stimulus conditions. Rows of conditions indicate constant visual stimulus, columns is constant audio stimulus. Model predictions is indicated by the red and blue dotted line.

### 3.1 Model comparison

To facilitate quantitative comparison with other models, we now fit the parameters of each model[2] to individual subject data, maximizing the likelihood of the model, i.e., the probability of the response frequencies under the model. The causal inference model fits human data better than the other models. Compared to the best fit of the causal inference model, the Bresciani model has a maximal log likelihood ratio (base $e$) of the data of $-22 \pm 6$ (mean $\pm$ s.e.m. over subjects), and the Roach model has a maximal log likelihood ratio of the data of $-18 \pm 6$. A causal inference model that maximizes the probability of being correct instead of minimizing the mean squared error has a maximal log likelihood ratio of $-18 \pm 3$. These values are considered decisive evidence in favor of the causal inference model that minimizes the mean squared error (for details, see [25]).

The parameter values found in the likelihood optimization of the causal model are as follows: $p_{\text{common}} = 0.28 \pm 0.05$, $\sigma_V = 2.14 \pm 0.22°$, $\sigma_A = 9.2 \pm 1.1°$, $\sigma_P = 12.3 \pm 1.1°$ (mean $\pm$ s.e.m. over subjects). We see that there is a relatively low prior probability of a common cause. In this paradigm, auditory localization is considerably less precise than visual localization. Also, there is a weak prior for central locations.

### 3.2 Localization bias

A useful quantity to gain more insight into the structure of multisensory data is the cross-modal bias. In our experiment, relative auditory bias is defined as the difference between the mean auditory estimate in a given condition and the real auditory position, divided by the difference between the real visual position and the real auditory position in this condition. If the influence of vision on the auditory estimate is strong, then the relative auditory bias will be high (close to one). It is well-known that bias decreases with spatial disparity and our experiment is no exception (solid line in Fig. 3; data were combined between positive and negative disparities). It can easily be shown that a traditional cue integration model would predict a bias equal to $\left(1 + \frac{\sigma_V^2}{\sigma_A^2}\right)^{-1}$, which would be close to 1 and independent of disparity, unlike the data. This shows that a mandatory integration model is an insufficient model of multisensory interactions.

We used the individual subject fittings from above and and averaged the auditory bias values obtained from those fits (i.e. we did not fit the bias data themselves). Fits are shown in Fig. 3 (dashed lines). We applied a paired t-test to the differences between the 5° and 20° disparity conditions (model-subject comparison). Using a double-sided test, the null hypothesis that the difference between the bias in the 5° and 20° conditions is correctly predicted by each model is rejected for the Bresciani model ($p < 0.002$) and the Roach model ($p < 0.042$) and accepted for the causal inference model ($p > 0.17$). Alternatively, with a single-sided test, the hypothesis is rejected for the Bresciani model ($p < 0.001$) and the Roach model ($p < 0.021$) and accepted for the causal inference model ($> 0.9$).

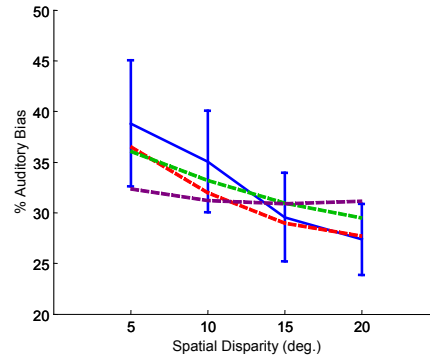

Figure 3: Auditory bias as a function of spatial disparity. Solid blue line: data. Red: Causal inference model. Green: Model by Roach et al. [23]. Purple: Model by Bresciani et al. [22]. Models were optimized on response frequencies (as in Fig. 2), not on the bias data.

The reason that the Bresciani model fares worst is that its prior distribution does not include a component that corresponds to independent causes. On

the contrary, the prior used in the Roach model contains two terms, one term that is independent of the disparity and one term that decreases with increasing disparity. It is thus functionally somewhat similar to the causal inference model.

## 4    Discussion

We have argued that any model of multisensory perception should account not only for situations of small, but also of large conflict. In these situations, segregation is more likely, in which the two stimuli are not perceived to have the same cause. Even when segregation occurs, the two stimuli can still influence each other.

We compared three Bayesian models designed to account for situations of large conflict by applying them to auditory-visual spatial localization data. We pointed out a common mistake: for non-Gaussian bisensory priors without mandatory integration, the response distribution can no longer be identified with the posterior distribution. After correct implementation of the three models, we found that the causal inference model is superior to the models with ad hoc bisensory priors. This is expected, as the nervous system actually needs to solve the problem of deciding which stimuli have a common cause and which stimuli are unrelated.

We have seen that multisensory perception is a suitable tool for studying causal inference. However, the causal inference model also has the potential to quantitatively explain a number of other perceptual phenomena, including perceptual grouping and binding, as well as within-modality cue combination [27, 28]. Causal inference is a universal problem: whenever the brain has multiple pieces of information it must decide if they relate to one another or are independent.

As the causal inference model describes how the brain processes probabilistic sensory information, the question arises about the neural basis of these processes. Neural populations encode probability distributions over stimuli through Bayes' rule, a type of coding known as probabilistic population coding. Recent work has shown how the optimal cue combination assuming a common cause can be implemented in probabilistic population codes through simple linear operations on neural activities [29]. This framework makes essential use of the structure of neural variability and leads to physiological predictions for activity in areas that combine multisensory input, such as the superior colliculus. Computational mechanisms for causal inference are expected have a neural substrate that generalizes these linear operations on population activities. A neural implementation of the causal inference model will open the door to a complete neural theory of multisensory perception.

## Footnotes

[1]This family of Bayesian posterior distributions also includes one used to successfully model cue combination in depth perception [27, 28]. In depth perception, however, there is no notion of segregation as always a single surface is assumed.

[2]The Roach et al. model has four free parameters ($\omega$, $\sigma_V$, $\sigma_A$, $\sigma_{\text{coupling}}$), the Bresciani et al. model has three ($\sigma_V$, $\sigma_A$, $\sigma_{\text{coupling}}$), and the causal inference model has four ($p_{\text{common}}$, $\sigma_V$, $\sigma_A$, $\sigma_P$). We do not consider the Shams et al. model here, since it has many more parameters and it is not immediately clear how in this model the erroneous identification of posterior with response distribution can be corrected.

## References

[1] H.L. Pick, D.H. Warren, and J.C. Hay. Sensory conflict in judgements of spatial direction. *Percept. Psychophys.*, 6:203205, 1969.

[2] D. H. Warren, R. B. Welch, and T. J. McCarthy. The role of visual-auditory "compellingness" in the ventriloquism effect: implications for transitivity among the spatial senses. *Percept Psychophys*, 30(6):557–64, 1981.

[3] D. Alais and D. Burr. The ventriloquist effect results from near-optimal bimodal integration. *Curr Biol*, 14(3):257–62, 2004.

[4] R. A. Jacobs. Optimal integration of texture and motion cues to depth. *Vision Res*, 39(21):3621–9, 1999.

[5] R. J. van Beers, A. C. Sittig, and J. J. Gon. Integration of proprioceptive and visual position-information: An experimentally supported model. *J Neurophysiol*, 81(3):1355–64, 1999.

[6] D. H. Warren and W. T. Cleaves. Visual-proprioceptive interaction under large amounts of conflict. *J Exp Psychol*, 90(2):206–14, 1971.

[7] C. E. Jack and W. R. Thurlow. Effects of degree of visual association and angle of displacement on the "ventriloquism" effect. *Percept Mot Skills*, 37(3):967–79, 1973.

[8] G. H. Recanzone. Auditory influences on visual temporal rate perception. *J Neurophysiol*, 89(2):1078–93, 2003.

[9] J. P. Bresciani, M. O. Ernst, K. Drewing, G. Bouyer, V. Maury, and A. Kheddar. Feeling what you hear: auditory signals can modulate tactile tap perception. *Exp Brain Res*, 162(2):172–80, 2005.

[10] R. Gepshtein, P. Leiderman, L. Genosar, and D. Huppert. Testing the three step excited state proton transfer model by the effect of an excess proton. *J Phys Chem A Mol Spectrosc Kinet Environ Gen Theory*, 109(42):9674–84, 2005.

[11] L. Shams, W. J. Ma, and U. Beierholm. Sound-induced flash illusion as an optimal percept. *Neuroreport*, 16(17):1923–7, 2005.

[12] G Thomas. Experimental study of the influence of vision on sound localisation. *J Exp Psychol*, 28:167177, 1941.

[13] W. R. Thurlow and C. E. Jack. Certain determinants of the "ventriloquism effect". *Percept Mot Skills*, 36(3):1171–84, 1973.

[14] C.S. Choe, R. B. Welch, R.M. Gilford, and J.F. Juola. The "ventriloquist effect": visual dominance or response bias. *Perception and Psychophysics*, 18:55–60, 1975.

[15] R. I. Bermant and R. B. Welch. Effect of degree of separation of visual-auditory stimulus and eye position upon spatial interaction of vision and audition. *Percept Mot Skills*, 42(43):487–93, 1976.

[16] R. B. Welch and D. H. Warren. Immediate perceptual response to intersensory discrepancy. *Psychol Bull*, 88(3):638–67, 1980.

[17] P. Bertelson and M. Radeau. Cross-modal bias and perceptual fusion with auditory-visual spatial discordance. *Percept Psychophys*, 29(6):578–84, 1981.

[18] P. Bertelson, F. Pavani, E. Ladavas, J. Vroomen, and B. de Gelder. Ventriloquism in patients with unilateral visual neglect. *Neuropsychologia*, 38(12):1634–42, 2000.

[19] D. A. Slutsky and G. H. Recanzone. Temporal and spatial dependency of the ventriloquism effect. *Neuroreport*, 12(1):7–10, 2001.

[20] J. Lewald, W. H. Ehrenstein, and R. Guski. Spatio-temporal constraints for auditory–visual integration. *Behav Brain Res*, 121(1-2):69–79, 2001.

[21] M. T. Wallace, G. E. Roberson, W. D. Hairston, B. E. Stein, J. W. Vaughan, and J. A. Schirillo. Unifying multisensory signals across time and space. *Exp Brain Res*, 158(2):252–8, 2004.

[22] J. P. Bresciani, F. Dammeier, and M. O. Ernst. Vision and touch are automatically integrated for the perception of sequences of events. *J Vis*, 6(5):554–64, 2006.

[23] N. W. Roach, J. Heron, and P. V. McGraw. Resolving multisensory conflict: a strategy for balancing the costs and benefits of audio-visual integration. *Proc Biol Sci*, 273(1598):2159–68, 2006.

[24] K. P. Kording and D. M. Wolpert. Bayesian decision theory in sensorimotor control. *Trends Cogn Sci*, 2006. 1364-6613 (Print) Journal article.

[25] K.P. Kording, U. Beierholm, W.J. Ma, S. Quartz, J. Tenenbaum, and L. Shams. Causal inference in multisensory perception. *PLoS ONE*, 2(9):e943, 2007.

[26] Z. Ghahramani. *Computational and psychophysics of sensorimotor integration*. PhD thesis, Massachusetts Institute of Technology, 1995.

[27] D. C. Knill. Mixture models and the probabilistic structure of depth cues. *Vision Res*, 43(7):831–54, 2003.

[28] D. C. Knill. Robust cue integration: A bayesian model and evidence from cue conflict studies with stereoscopic and figure cues to slant. *Journal of Vision*, 7(7):2–24.

[29] W. J. Ma, J. M. Beck, P. E. Latham, and A. Pouget. Bayesian inference with probabilistic population codes. *Nat Neurosci*, 9(11):1432–8, 2006.

